# Layered Dynamic Textures

**Antoni B. Chan**     **Nuno Vasconcelos**
Department of Electrical and Computer Engineering
University of California, San Diego
abchan@ucsd.edu,  nuno@ece.ucsd.edu

## Abstract

A dynamic texture is a video model that treats a video as a sample from a spatio-temporal stochastic process, specifically a linear dynamical system. One problem associated with the dynamic texture is that it cannot model video where there are multiple regions of distinct motion. In this work, we introduce the layered dynamic texture model, which addresses this problem. We also introduce a variant of the model, and present the EM algorithm for learning each of the models. Finally, we demonstrate the efficacy of the proposed model for the tasks of segmentation and synthesis of video.

## 1  Introduction

Traditional motion representations, based on optical flow, are inherently local and have significant difficulties when faced with aperture problems and noise. The classical solution to this problem is to regularize the optical flow field [1, 2, 3, 4], but this introduces undesirable smoothing across motion edges or regions where the motion is, by definition, not smooth (e.g. vegetation in outdoors scenes). More recently, there have been various attempts to model video as a superposition of layers subject to homogeneous motion. While layered representations exhibited significant promise in terms of combining the advantages of regularization (use of global cues to determine local motion) with the flexibility of local representations (little undue smoothing), this potential has so far not fully materialized. One of the main limitations is their dependence on parametric motion models, such as affine transforms, which assume a piece-wise planar world that rarely holds in practice [5, 6]. In fact, layers are usually formulated as "cardboard" models of the world that are warped by such transformations and then stitched to form the frames in a video stream [5]. This severely limits the types of video that can be synthesized: while layers showed most promise as models for scenes composed of ensembles of objects subject to homogeneous motion (e.g. leaves blowing in the wind, a flock of birds, a picket fence, or highway traffic), very little progress has so far been demonstrated in actually modeling such scenes.

Recently, there has been more success in modeling complex scenes as *dynamic textures* or, more precisely, samples from stochastic processes defined over space and time [7, 8, 9, 10]. This work has demonstrated that modeling both the dynamics and appearance of video as stochastic quantities leads to a much more powerful generative model for video than that of a "cardboard" figure subject to parametric motion. In fact, the dynamic texture model has shown a surprising ability to abstract a wide variety of complex patterns of motion and appearance into a *simple* spatio-temporal model. One major current limitation

of the dynamic texture framework, however, is its inability to account for visual processes consisting of *multiple, co-occurring, dynamic textures*. For example, a flock of birds flying in front of a water fountain, highway traffic moving at different speeds, video containing both trees in the background and people in the foreground, and so forth. In such cases, the existing dynamic texture model is inherently incorrect, since it must represent multiple motion fields with a single dynamic process.

In this work, we address this limitation by introducing a new generative model for video, which we denote by the *layered dynamic texture* (LDT). This consists of augmenting the dynamic texture with a discrete *hidden* variable, that enables the assignment of different dynamics to different regions of the video. Conditioned on the state of this hidden variable, the video is then modeled as a simple dynamic texture. By introducing a shared dynamic representation for all the pixels in the same region, the new model is a layered representation. When compared with traditional layered models, it replaces the process of layer formation based on "warping of cardboard figures" with one based on sampling from the generative model (for both dynamics and appearance) provided by the dynamic texture. This enables a much richer video representation. Since each layer is a dynamic texture, the model can also be seen as a multi-state dynamic texture, which is capable of assigning different dynamics and appearance to different image regions.

We consider two models for the LDT, that differ in the way they enforce consistency of layer dynamics. One model enforces stronger consistency but has no closed-form solution for parameter estimates (which require sampling), while the second enforces weaker consistency but is simpler to learn. The models are applied to the segmentation and synthesis of sequences that are challenging for traditional vision representations. It is shown that stronger consistency leads to superior performance, demonstrating the benefits of sophisticated layered representations. The paper is organized as follows. In Section 2, we introduce the two layered dynamic texture models. In Section 3 we present the EM algorithm for learning both models from training data. Finally, in Section 4 we present an experimental evaluation in the context of segmentation and synthesis.

## 2 Layered dynamic textures

We start with a brief review of dynamic textures, and then introduce the layered dynamic texture model.

### 2.1 Dynamic texture

A dynamic texture [7] is a generative model for video, based on a linear dynamical system. The basic idea is to separate the visual component and the underlying dynamics into two processes. While the dynamics are represented as a time-evolving state process $x_t \in \mathbb{R}^n$, the appearance of frame $y_t \in \mathbb{R}^N$ is a linear function of the current state vector, plus some observation noise. Formally, the system is described by

$$\begin{cases} x_t = Ax_{t-1} + Bv_t \\ y_t = Cx_t + \sqrt{r}w_t \end{cases} \tag{1}$$

where $A \in \mathbb{R}^{n \times n}$ is a transition matrix, $C \in \mathbb{R}^{N \times n}$ a transformation matrix, $Bv_t \sim_{iid} \mathcal{N}(0, Q,)$ and $\sqrt{r}w_t \sim_{iid} \mathcal{N}(0, rI_N)$ the state and observation noise processes parameterized by $B \in \mathbb{R}^{n \times n}$ and $r \in \mathbb{R}$, and the initial state $x_0 \in \mathbb{R}^n$ is a constant. One interpretation of the dynamic texture model is that the columns of $C$ are the principal components of the video frames, and the state vectors the PCA coefficients for each video frame. This is the case when the model is learned with the method of [7].

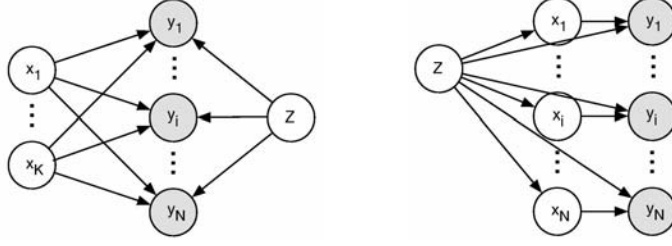

Figure 1: The layered dynamic texture (left), and the approximate layered dynamic texture (right). $y_i$ is an observed pixel over time, $x_j$ is a hidden state process, and $Z$ is the collection of layer assignment variables $z_i$ that assigns each pixels to one of the state processes.

An alternative interpretation considers a single pixel as it evolves over time. Each coordinate of the state vector $x_t$ defines a one-dimensional random trajectory in time. A pixel is then represented as a weighted sum of random trajectories, where the weighting coefficients are contained in the corresponding row of $C$. This is analogous to the discrete Fourier transform in signal processing, where a signal is represented as a weighted sum of complex exponentials although, for the dynamic texture, the trajectories are not necessarily orthogonal. This interpretation illustrates the ability of the dynamic texture to model the same motion under different intensity levels (e.g. cars moving from the shade into sunlight) by simply scaling the rows of $C$. Regardless of interpretation, the simple dynamic texture model has only one state process, which restricts the efficacy of the model to video where the motion is homogeneous.

## 2.2 Layered dynamic textures

We now introduce the *layered dynamic texture* (LDT), which is shown in Figure 1 (left). The model addresses the limitations of the dynamic texture by relying on a set of state processes $X = \{x^{(j)}\}_{j=1}^{K}$ to model different video dynamics. The layer assignment variable $z_i$ assigns pixel $y_i$ to one of the state processes (layers), and conditioned on the layer assignments, the pixels in the same layer are modeled as a dynamic texture. In addition, the collection of layer assignments $Z = \{z_i\}_{i=1}^{N}$ is modeled as a Markov random field (MRF) to ensure spatial layer consistency. The linear system equations for the layered dynamic texture are

$$\begin{cases} x_t^{(j)} = A^{(j)} x_{t-1}^{(j)} + B^{(j)} v_t^{(j)} & j \in \{1, \cdots, K\} \\ y_{i,t} = C_i^{(z_i)} x_t^{(z_i)} + \sqrt{r^{(z_i)}} w_{i,t} & i \in \{1, \cdots, N\} \end{cases} \tag{2}$$

where $C_i^{(j)} \in \mathbb{R}^{1 \times n}$ is the transformation from the hidden state to the observed pixel domain for each pixel $y_i$ and each layer $j$, the noise parameters are $B^{(j)} \in \mathbb{R}^{n \times n}$ and $r^{(j)} \in \mathbb{R}$, the iid noise processes are $w_{i,t} \sim_{iid} \mathcal{N}(0,1)$ and $v_t^{(j)} \sim_{iid} \mathcal{N}(0, I_n)$, and the initial states are drawn from $x_1^{(j)} \sim \mathcal{N}(\mu^{(j)}, S^{(j)})$. As a generative model, the layered dynamic texture assumes that the state processes $X$ and the layer assignments $Z$ are independent, i.e. layer motion is independent of layer location, and vice versa. As will be seen in Section 3, this makes the expectation-step of the EM algorithm intractable to compute in closed-form. To address this issue, we also consider a slightly different model.

## 2.3 Approximate layered dynamic texture

We now consider a different model, the approximate layered dynamic texture (ALDT), shown in Figure 1 (right). Each pixel $y_i$ is associated with its own state process $x_i$, and a

different dynamic texture is defined for each pixel. However, dynamic textures associated with the same layer share the same set of dynamic parameters, which are assigned by the layer assignment variable $z_i$. Again, the collection of layer assignments $Z$ is modeled as an MRF but, unlike the first model, conditioning on the layer assignments makes all the pixels independent. The model is described by the following linear system equations

$$\begin{cases} x_{i,t} = A^{(z_i)} x_{i,t-1} + B^{(z_i)} v_{i,t} & i \in \{1, \cdots, N\} \\ y_{i,t} = C_i^{(z_i)} x_{i,t} + \sqrt{r^{(z_i)}} w_{i,t} \end{cases} \tag{3}$$

where the noise processes are $w_{i,t} \sim_{iid} \mathcal{N}(0,1)$ and $v_{i,t} \sim_{iid} \mathcal{N}(0, I_n)$, and the initial states are given by $x_{i,1} \sim \mathcal{N}(\mu^{(z_i)}, S^{(z_i)})$. This model can also be seen as a video extension of the popular image MRF models [11], where class variables for each pixel form an MRF grid and each class (e.g. pixels in the same segment) has some class-conditional distribution (in our case a linear dynamical system).

The main difference between the two proposed models is in the enforcement of consistency of dynamics within a layer. With the LDT, consistency of dynamics is strongly enforced by requiring each pixel in the layer to be associated with the *same* state process. On the other hand, for the ALDT, consistency within a layer is weakly enforced by allowing the pixels to be associated with *many* instantiations of the state process (instantiations associated with the same layer sharing the same dynamic parameters). This weaker dependency structure enables a more efficient learning algorithm.

### 2.4 Modeling layer assignments

The MRF which determines layer assignments has the following distribution

$$p(Z) = \frac{1}{\mathcal{Z}} \prod_i \psi_i(z_i) \prod_{(i,j)\in\mathcal{E}} \psi_{i,j}(z_i, z_j) \tag{4}$$

where $\mathcal{E}$ is the set of edges in the MRF grid, $\mathcal{Z}$ a normalization constant (partition function), and $\psi_i$ and $\psi_{i,j}$ potential functions of the form

$$\psi_i(z_i) = \begin{cases} \alpha_1 & , z_i = 1 \\ \vdots & \vdots \\ \alpha_K & , z_i = K \end{cases} \qquad \psi_{i,j}(z_i, z_j) = \begin{cases} \gamma_1 & , z_i = z_j \\ \gamma_2 & , z_i \neq z_j \end{cases} \tag{5}$$

The potential function $\psi_i$ defines a prior likelihood for each layer, while $\psi_{i,j}$ attributes higher probability to configurations where neighboring pixels are in the same layer. While the parameters for the potential functions could be learned for each model, we instead treat them as constants that can be estimated from a database of manually segmented training video.

## 3   Parameter estimation

The parameters of the model are learned using the Expectation-Maximization (EM) algorithm [12], which iterates between estimating hidden state variables $X$ and hidden layer assignments $Z$ from the current parameters, and updating the parameters given the current hidden variable estimates. One iteration of the EM algorithm contains the following two steps

- E-Step: $\mathcal{Q}(\Theta; \hat{\Theta}) = \mathrm{E}_{X,Z|Y;\hat{\Theta}}(\log p(X, Y, Z; \Theta))$
- M-Step: $\hat{\Theta}^* = \mathrm{argmax}_\Theta \, \mathcal{Q}(\Theta; \hat{\Theta})$

In the remainder of this section, we briefly describe the EM algorithm for the two proposed models. Due to the limited space available, we refer the reader to the companion technical report [13] for further details.

## 3.1   EM for the layered dynamic texture

The E-step for the layered dynamic texture computes the conditional mean and covariance of $x_t^{(j)}$ given the observed video $Y$. These expectations are intractable to compute in closed-form since it is not known to which state process each of the pixels $y_i$ is assigned, and it is therefore necessary to marginalize over all configurations of $Z$. This problem also appears for the computation of the posterior layer assignment probability $p(z_i = j|Y)$. The method of approximating these expectations which we currently adopt is to simply average over draws from the posterior $p(X, Z|Y)$ using a Gibbs sampler. Other approximations, e.g. variational methods or belief propagation, could be used as well. We plan to consider them in the future. Once the expectations are known, the M-step parameter updates are analogous to those required to learn a regular linear dynamical system [15, 16], with a minor modification in the updates if the transformation matrices $C_i^{(j)}$. See [13] for details.

## 3.2   EM for the approximate layered dynamic texture

The ALDT model is similar to the mixture of dynamic textures [14], a video clustering model that treats a collection of videos as a sample from a collection of dynamic textures. Since, for the ALDT model, each pixel is sampled from a set of one-dimensional dynamic textures, the EM algorithm is similar to that of the mixture of dynamic textures. There are only two differences. First, the E-step computes the posterior assignment probability $p(z_i|Y)$ given all the observed data, rather than conditioned on a single data point $p(z_i|y_i)$. The posterior $p(z_i|Y)$ can be approximated by sampling from the full posterior $p(Z|Y)$ using Markov-Chain Monte Carlo [11], or with other methods, such as loopy belief propagation. Second, the transformation matrix $C_i^{(j)}$ is different for each pixel, and the E and M steps must be modified accordingly. Once again, the details are available in [13].

# 4   Experiments

In this section, we show the efficacy of the proposed model for segmentation and synthesis of several videos with multiple regions of distinct motion. Figure 2 shows the three video sequences used in testing. The first (top) is a composite of three distinct video textures of water, smoke, and fire. The second (middle) is of laundry spinning in a dryer. The laundry in the bottom left of the video is spinning in place in a circular motion, and the laundry around the outside is spinning faster. The final video (bottom) is of a highway [17] where the traffic in each lane is traveling at a different speed. The first, second and fourth lanes (from left to right) move faster than the third and fifth. All three videos have multiple regions of motion and are therefore properly modeled by the models proposed in this paper, but not by a regular dynamic texture.

Four variations of the video models were fit to each of the three videos. The four models were the layered dynamic texture and the approximate layered dynamic texture models (LDT and ALDT), and those two models without the MRF layer assignment (LDT-iid and ALDT-iid). In the latter two cases, the layers assignments $z_i$ are distributed as iid multinomials. In all the experiments, the dimension of the state space was $n = 10$. The MRF grid was based on the eight-neighbor system (with cliques of size 2), and the parameters of the potential functions were $\gamma_1 = 0.99$, $\gamma_2 = 0.01$, and $\alpha_j = 1/K$. The expectations required by the EM algorithm were approximated using Gibbs sampling for the LDT and LDT-iid models and MCMC for the ALDT model. We first present segmentation results, to show

that the models can effectively separate layers with different dynamics, and then discuss results relative to video synthesis from the learned models.

## 4.1 Segmentation

The videos were segmented by assigning each of the pixels to the most probable layer conditioned on the observed video, i.e.

$$z_i^* = \operatorname*{argmax}_j p(z_i = j | Y) \qquad (6)$$

Another possibility would be to assign the pixels by maximizing the posterior of all the pixels $p(Z|Y)$. While this maximizes the true posterior, in practice we obtained similar results with the two methods. The former method was chosen because the individual posterior distributions are already computed during the E-step of EM.

The columns of Figure 3 show the segmentation results obtained with for the four models: LDT and LDT-iid in columns (a) and (b), and ALDT and ALDT-iid in columns (c) and (d). The segmented video is also available at [18]. From the segmentations produced by the iid models, it can be concluded that the composite and laundry videos can be reasonably well segmented without the MRF prior. This confirms the intuition that the various video regions contain very distinct dynamics, which can only be modeled with separate state processes. Otherwise, the pixels should be either randomly assigned among the various layers, or uniformly assigned to one of them. The segmentations of the traffic video using the iid models are poor. While the dynamics are different, the differences are significantly more subtle, and segmentation requires stronger enforcement of layer consistency. In general, the segmentations using LDT-iid are better than to those of the ALDT-iid, due to the weaker form of layer consistency imposed by the ALDT model. While this deficiency is offset by the introduction of the MRF prior, the stronger consistency enforced by the LDT model always results in better segmentations. This illustrates the need for the design of sophisticated layered representations when the goal is to model video with subtle inter-layer variations. As expected, the introduction of the MRF prior improves the segmentations produced by both models. For example, in the composite sequence all erroneous segments in the water region are removed, and in the traffic sequence, most of the speckled segmentation also disappears.

In terms of the overall segmentation quality, both LDT and ALDT are able to segment the composite video perfectly. The segmentation of the laundry video by both models is plausible, as the laundry tumbling around the edge of the dryer moves faster than that spinning in place. The two models also produce reasonable segmentations of the traffic video, with the segments roughly corresponding to the different lanes of traffic. Much of the errors correspond to regions that either contain intermittent motion (e.g. the region between the lanes) or almost no motion (e.g. truck in the upper-right corner and flat-bed truck in the third lane). Some of these errors could be eliminated by filtering the video before segmentation, but we have attempted no pre or post-processing. Finally, we note that the laundry and traffic videos are not trivial to segment with standard computer vision techniques, namely methods based on optical flow. This is particularly true in the case of the traffic video where the abundance of straight lines and flat regions makes computing the correct optical flow difficult due to the aperture problem.

## 4.2 Synthesis

The layered dynamic texture is a generative model, and hence a video can be synthesized by drawing a sample from the learned model. A synthesized composite video using the LDT, ALDT, and the normal dynamic texture can be found at [18]. When modeling a video with multiple motions, the regular dynamic texture will average different dynamics.

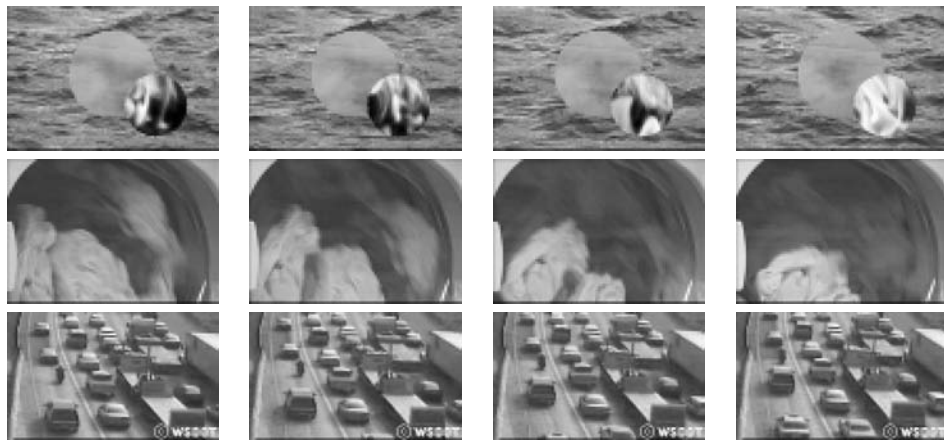

Figure 2: Frames from the test video sequences: (top) composite of water, smoke, and fire video textures; (middle) spinning laundry in a dryer; and (bottom) highway traffic with lanes traveling at different speeds.

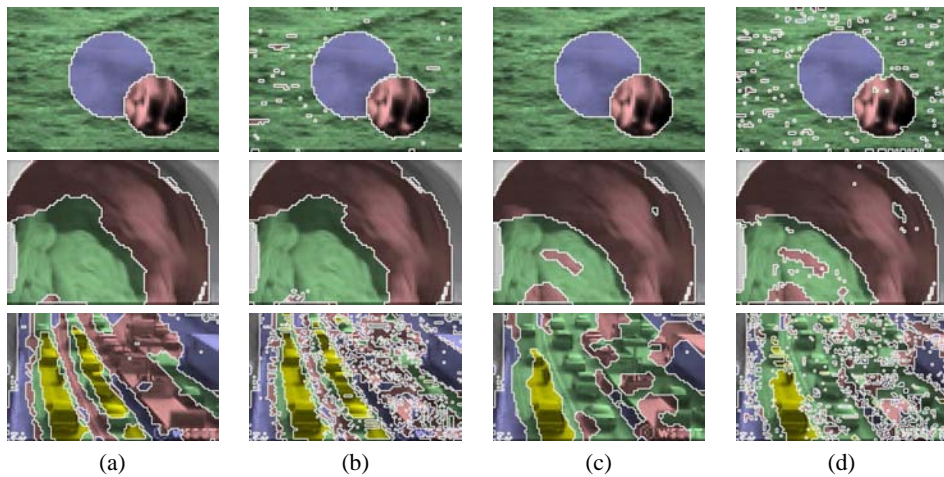

(a)          (b)          (c)          (d)

Figure 3: Segmentation results for each of the test videos using: (a) the layered dynamic texture, and (b) the layered dynamic texture without MRF; (c) the approximate layered dynamic texture, and (d) the approximate LDT without MRF.

This is noticeable in the synthesized video, where the fire region does not flicker at the same speed as in the original video. Furthermore, the motions in different regions are coupled, e.g. when the fire begins to flicker faster, the water region ceases to move smoothly. In contrast, the video synthesized from the layered dynamic texture is more realistic, as the fire region flickers at the correct speed, and the different regions follow their own motion patterns. The video synthesized from the ALDT appears noisy because the pixels evolve from different instantiations of the state process. Once again this illustrates the need for sophisticated layered models.

## References

[1] B. K. P. Horn. *Robot Vision*. McGraw-Hill Book Company, New York, 1986.

[2] B. Horn and B. Schunk. Determining optical flow. *Artificial Intelligence*, vol. 17, 1981.

[3] B. Lucas and T. Kanade. An iterative image registration technique with an application to stereo vision. *Proc. DARPA Image Understanding Workshop*, 1981.

[4] J. Barron, D. Fleet, and S. Beauchemin. Performance of optical flow techniques. *International Journal of Computer Vision*, vol. 12, 1994.

[5] J. Wang and E. Adelson. Representing moving images with layers. *IEEE Trans. on Image Processing*, vol. 3, September 1994.

[6] B. Frey and N. Jojic. Estimating mixture models of images and inferring spatial transformations using the EM algorithm. In *IEEE Conference on Computer Vision and Pattern Recognition*, 1999.

[7] G. Doretto, A. Chiuso, Y. N. Wu, and S. Soatto. Dynamic textures. *International Journal of Computer Vision*, vol. 2, pp. 91-109, 2003.

[8] G. Doretto, D. Cremers, P. Favaro, and S. Soatto. Dynamic texture segmentation. In *IEEE International Conference on Computer Vision*, vol. 2, pp. 1236-42, 2003.

[9] P. Saisan, G. Doretto, Y. Wu, and S. Soatto. Dynamic texture recognition. In *IEEE Conference on Computer Vision and Pattern Recognition, Proceedings*, vol. 2, pp. 58-63, 2001.

[10] A. B. Chan and N. Vasconcelos. Probabilistic kernels for the classification of auto-regressive visual processes. In *IEEE Conference on Computer Vision and Pattern Recognition*, vol. 1, pp. 846-51, 2005.

[11] S. Geman and D. Geman. Stochastic relaxation, Gibbs distribution, and the Bayesian restoration of images. *IEEE Transactions on Pattern Analysis and Machine Intelligence*, vol. 6(6), pp. 721-41, 1984.

[12] A. P. Dempster, N. M. Laird, and D. B. Rubin. Maximum likelihood from incomplete data via the EM algorithm. *Journal of the Royal Statistical Society B*, vol. 39, pp. 1-38, 1977.

[13] A. B. Chan and N. Vasconcelos. The EM algorithm for layered dynamic textures. *Technical Report SVCL-TR-2005-03*, June 2005. http://www.svcl.ucsd.edu/.

[14] A. B. Chan and N. Vasconcelos. Mixtures of dynamic textures. In *IEEE International Conference on Computer Vision*, vol. 1, pp. 641-47, 2005.

[15] R. H. Shumway and D. S. Stoffer. An approach to time series smoothing and forecasting using the EM algorithm. *Journal of Time Series Analysis*, vol. 3(4), pp. 253-64, 1982.

[16] S. Roweis and Z. Ghahramani. A unifying review of linear Gaussian models. *Neural Computation*, vol. 11, pp. 305-45, 1999.

[17] http://www.wsdot.wa.gov

[18] http://www.svcl.ucsd.edu/~abc/nips05/
